# Temporal Dynamics of Cognitive Control

**Jeremy R. Reynolds**
Department of Psychology
University of Denver
Denver, CO 80208
jeremy.reynolds@psy.du.edu

**Michael C. Mozer**
Department of Computer Science and
Institute of Cognitive Science
University of Colorado
Boulder, CO 80309
mozer@colorado.edu

## Abstract

Cognitive control refers to the flexible deployment of memory and attention in response to task demands and current goals. Control is often studied experimentally by presenting sequences of stimuli, some demanding a response, and others modulating the stimulus-response mapping. In these tasks, participants must maintain information about the current stimulus-response mapping in working memory. Prominent theories of cognitive control use recurrent neural nets to implement working memory, and optimize memory utilization via reinforcement learning. We present a novel perspective on cognitive control in which working memory representations are intrinsically probabilistic, and control operations that maintain and update working memory are dynamically determined via probabilistic inference. We show that our model provides a parsimonious account of behavioral and neuroimaging data, and suggest that it offers an elegant conceptualization of control in which behavior can be cast as optimal, subject to limitations on learning and the rate of information processing. Moreover, our model provides insight into how task instructions can be directly translated into appropriate behavior and then efficiently refined with subsequent task experience.

## 1   Introduction

Cognitive control can be characterized as the ability to guide behavior according to current goals and plans. Control often involves overriding default or overlearned behaviors. Classic examples of experimental tasks requiring this ability include Stroop, Wisconsin card sorting, and task switching (for a review, see [1]). Although these paradigms vary in superficial features, they share the key underlying property that successful performance involves updating and maintaining a *task set*. The task set holds the information required for successful performance, e.g., the stimulus-response mapping, or the dimension along which stimuli are to be classified or reported. For example, in Wisconsin card sorting, participants are asked to classify cards with varying numbers of instances of a colored symbol. The classification might be based on color, symbol, or numerosity; instructions require participants to identify the current dimension through trial and error, and perform the appropriate classification until the dimension switches after some unspecified number of trials. Thus, it requires participants to maintain a task set—the classification dimension—in working memory (WM). Likewise, in the Stroop task, stimuli are color names presented in various ink colors, and the task set specifies whether the color is to be named or the word is to be read.

To understand cognitive control, we need to characterize the brain's policy for updating, maintaining, and utilizing task set. Moreover, we need to develop theories of how task instructions are translated into a policy, and how this policy is refined with subsequent experience performing a task.

## 1.1 Current Computational Theories of Control

From a purely computational perspective, control is not a great challenge. Every computer program modulates its execution based on internal state variables. The earliest psychological theories of control had this flavor: Higher cognitive function was conceived of as a logical symbol system whose variables could be arbitrarily bound [2], allowing for instructions to be used appropriately—and perfectly—to update representations that support task performance. For example, in the Wisconsin card sorting task, the control instruction—the classification dimension—would be bound to a variable, and responses would be produced by rules of the form, "If the current dimension is D and the stimulus is X, respond Y". Behavioral data indicate that this naive computational perspective is unlikely to be how control is implemented in the brain. Consider the following phenomena:

- When participants are asked to switch tasks, performance on the first trial following a switch is inefficient, although performance on subsequent trials is efficient, suggesting that loading a new task set depends on actually performing the new task [3]. This finding is observed even for very simple tasks, and even when the switches are regular, highly predictable, and well practiced.

- Switch costs are asymmetric, such that switching from an easy task to a difficult task is easier than vice-versa [4].

- Some task sets are more difficult to implement than others. For example, in the Stroop task, reading the word is quick and accurate, but naming the ink color is not [5].

- The difficulty of a particular task depends not only on the characteristics of the task itself, but also on context in which participants might be called upon to perform [6].

To account for phenomena such as these, theories of control have in recent years focused on how control can be implemented in cortical neural networks. In the prevailing neural-network-based theory, task set is represented in an activity-based memory system, i.e., a population of neurons whose recurrent activity maintains the representation over time. This active memory, posited to reside in prefrontal cortex (PFC), serves to bias ongoing processing in posterior cortical regions to achieve flexibility and arbitrary, task-dependent stimulus-response mappings (for review, see [1]). For example, in the Stroop task, instructions to report the ink color might bias the neural population representing colors—i.e., increase their baseline activity prior to stimulus onset—such that when stimulus information arrives, it will reach threshold more rapidly, and will beat out the neural population that represents word orthography in triggering response systems [7]. In this framework, a control policy must specify the updating and maintenance task set, which involves when to gate new representations into WM and the strength of the recurrent connection that maintains the memory. Further, the policy must specify which WM populations bias which posterior representations, and the degree to which biasing is required. Some modelers have simply specified the policy by hand [8], whereas most pretrain the model to perform a task—in a manner meant to reflect long-term learning prior to experimental testing [7, 9, 10].

These models provide an account for a range of neurophysiological and behavioral data. However, they might be criticized on a number of grounds. First, like their symbolic predecessors, the neural network models must often be crippled arbitrarily to explain data; for example, by limiting the strength of recurrent memory connections, the models obtain task set decay and can explain error data. Second, the models require a stage of training which is far more akin to how a monkey learns to perform a task than to how people follow task instructions. The reinforcement-learning based models require a long stage of trial-and-error learning before the appropriate control policy emerges. Whereas monkeys are often trained for months prior to testing, a notable characteristic of humans is that they can perform a task adequately on the first trial from task instructions [11].

## 2 Control as Inference

Our work aims to provide an alternative, principled conceptualization of cognitive control. Our goal is to develop an elegant theoretical framework with few free parameters that can easily be applied to a wide range of experimental tasks. With strong computational and algorithmic constraints, our framework has few degrees of freedom, and consequently, makes strong, experimentally verifiable

predictions. Additionally, as a more abstract framework than the neural net theories, one aim is to provide insight as to how task instructions can be used directly and immediately to control behavior.

A fundamental departure of our approach from previous approaches is to consider WM as inherently probabilistic. That is, instead of proposing that task set is stored in an all-or-none fashion, we wish to allow for task set—as well as all cortical representations—to be treated as random variables. This notion is motivated by computational neuroscience models showing how population codes can be used to compute under uncertainty [12].

Given inherently probabilistic representations, it is natural to treat the problems of task set updating, maintenance and utilization as probabilistic inference. To provide an intuition about our approach, consider this scenario. I will walk around my house and tell you what objects I see. Your job is to guess what I'll report next. Suppose I report the following sequence: REFRIGERATOR, STOVE, SINK, TOILET, SHOWER, DRESSER. To guess what I'm likely to see next, you need to infer what room I am in. Even though the room is a latent variable, it can be inferred from the sequence of observations. At some points in the sequence, the room can be determined with great confidence (e.g., after seeing TOILET and SHOWER). At other times, the room is ambiguous (e.g., following SINK), and only weak inferences can be drawn.

By analogy, our approach to cognitive control treats task set as a latent variable that must be inferred from observations. The observations consist of stimulus-response-feedback triples.Sometimes the observations will strongly constrain the task set, as in the Stroop task when the word GREEN is shown in color red, and the correct response is red, or when an explicit instruction is given to report the ink color; but other times the observations provide little constraint, as when the word RED is shown in color red, and the correct response is red. One inference problem is therefore to determine task set from the stimulus-response sequence. A second, distinct inference problem is to determine the correct response on the current trial from the current stimulus and the trial history. Thus, in our approach, control and response selection are cast as inference under uncertainty.

In this paper, we flesh out a model based on this approach. We use the model to account for behavioral data from two experiments. Each experiment involves a complex task environment in which experimental participants are required to switch among eight tasks that have different degrees of overlap and inconsistency with one another. Having constrained the model by fitting behavioral data, we then show that the model can explain neuroimaging data. Moreover, the model provides a different interpretation to these data than has been suggested previously. Beyond accounting for data, the model provides an elegant theoretical framework in which control and response selection can be cast as optimal, subject to limitations on the processing architecture.

## 3   Methods

Our model addresses data from two experiments conducted by Koechlin, Ody, and Kouneiher [6]. In each experiment, participants are shown blocks of 12 trials, preceded by a cue that indicates which of the eight tasks is to be performed with the stimuli in that block. The task specifies a stimulus-response mapping. The stimuli in Experiments 1 and 2 are colored squares and colored letters, respectively. Examples of the sequence of cues and stimuli for the two experiments is shown in Figure 1A. In both experiments, there are two potential responses.

The stimulus-response mappings for Experiment 1 are shown in the eight numbered boxes of Figure 1C. (The layout of the boxes will be explained shortly.) Consider task 3 in the upper left corner of the Figure. The notation indicates that task 3 requires a left response to the green square, a right response to a red square, and no response (hereafter, *no-go*) to a white square. Task 4 is identical to task 3, and the duplication is included because the tasks are described as distinct to participants and each is associated with a unique task cue. The duplication makes the stimulus-response mapping twice as likely, because the eight tasks have uniform priors. Task 1 (lower left corner of the figure) requires a left response for a green square and no-go for a white square. There are no red stimuli in the task 1 blocks, and the green→left mapping is depicted twice to indicate that the probability of a green square appearing in the block is twice that of a white square.

We now explain the $3 \times 2$ arrangement of cells in Figure 1C. First the rows. The four tasks in the lower row allow for only one possible response (not counting no-go as a response), whereas the four tasks in the upper row demand that a choice be made between two possible responses.

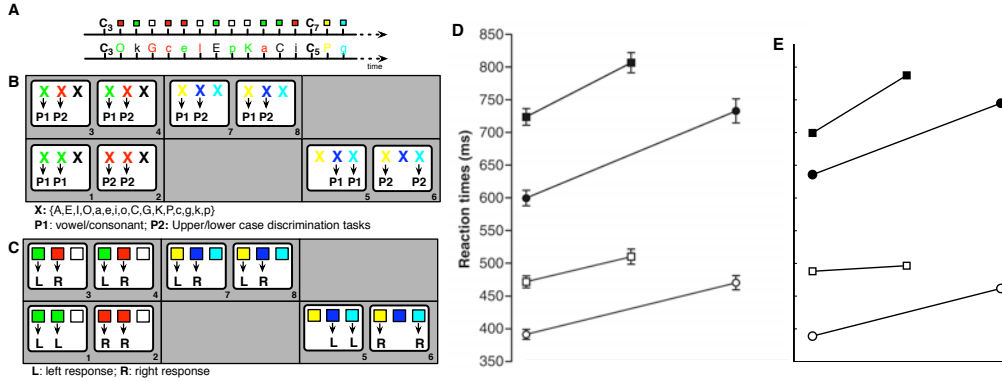

Figure 1: (A) Examples of stimulus sequences from Exp. 1 and 2 (top and bottom arrows, respectively) of [6]. (B) Eight tasks in Exp. 2, adapted from [6]. (C) Eight tasks in Exp. 1. (D) Response times from participants in Exp. 1 and 2 (white and black points, respectively). The data points correspond to the filled grey cells of (B) and (C), and appear in homologous locations. X-axis of graph corresponds to columns of the 3×2 array of cells in (B) and (C); squares and circles correspond to top and bottom row of each 3×2 array. (E) Simulation results from the model.

Thus, the two rows differ in terms of the demands placed on *response selection*. The three columns differ in the *importance of the task identity*. In the leftmost column, task identity does not matter, because each mapping (e.g., green→left) is consistent irrespective of the task identity. In contrast, tasks utilizing yellow, blue, and cyan stimuli involve varied mappings. For example, yellow maps to left in two tasks, to right in one task, and to no-go in one task. The tasks in the middle column are somewhat less dependent on task identity, because the stimulus-response mappings called for have the highest prior. Thus, the three columns represent a continuum along which the importance of task identity varies, from being completely irrelevant (left column) to being critical for correct performance (right column). Empty cells within the grid are conceptually possible, but were omitted from the experiment.

Experiment 2 has the same structure as Experiment 1 (Figure 1B), with an extra level of complexity. Rather than mapping a color to a response, the color determines which property of the stimulus is to be used to select a response. For example, task 3 of Figure 1B demands that a green letter stimulus (denoted as X here) be classified as a vowel or consonant (property P1), whereas a red letter stimulus be classified as upper or lower case (property P2). Thus, Experiment 2 places additional demands of *stimulus classification and selection of the appropriate stimulus dimension*.

Participants in each experiment received extensive practice on the eight tasks before being tested. Testing involved presenting each task following each other task, for a total of 64 test blocks.

## 3.1 A Probabilistic Generative Model of Control Tasks

Following the style of many probabilistic models in cognitive science, we have designed a generative model of the domain, and then invert the model to perform recognition via Bayesian inference. In our case, the generative model is of the control task, i.e., the model produces sequences of stimulus-response pairs such that the actual trial sequence would be generated with high probability. Instead of learning this model from data, though, we assume that task instructions are 'programmed' into the model.

Our generative model of control tasks is sketched in Figure 2A as a dynamical Bayes net. Vertical slices of the model represent the trial sequence, with the subscript denoting the trial index. First we explain the nodes and dependencies and then describe the conditional probability distributions (CPDs).

The $B$ node represents the task associated with the current block of trials. (We use the term 'block' as shorthand notation for this task.) The block on trial $k$ has 8 possible values in the experiments we

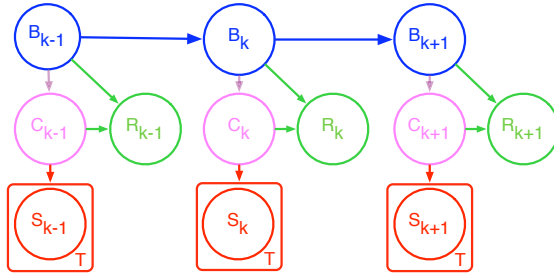

Figure 2: Dynamical Bayes net depiction of our generative model of control tasks, showing the trial-to-trial structure of the model.

model, and its value depends on the block on trial $k-1$. The block determines the category of the stimulus, $C$, which in turn determines the stimulus identity, $S$. The categories relevant to the present experiments are: color label, block cue (the cue that identifies the task in the next block), upper/lower case for letters, and consonant/vowel for letters. The stimuli correspond to instantiations of these categories, e.g., the letter $Q$ which is an instance of an upper case consonant. Finally, the $R$ node denotes the response, which depends both on the current stimulus category and the current block.

This description of the model is approximate for two reasons. First, we decompose the category and stimulus representations into shape and color dimensions, expanding $C$ into $C^{\text{color}}$ and $C^{\text{shape}}$, and $S$ into $S^{\text{color}}$ and $S^{\text{shape}}$. (When we refer to $C$ or $S$ without the superscript, it will denote both the shape and color components.) Second, we wish to model the temporal dynamics of a single trial, in order to explain response latencies. Although one could model the temporal dynamics as part of the dynamical Bayes net architecture, we adopted a simpler and nearly equivalent approach, which is to explicitly represent time, $T$, within a trial, and to assume that in the generative model, stimulus information accumulates exponentially over time. With normalization of probabilities, this formulation is identical to a naive Bayes model with conditionally independent stimulus observations at each time step. With these two modifications, the slices of the network (indicated by the dashed rectangle in Figure 2A) are as depicted in Figure 2B.

To this point, we've designed a generic model of any experimental paradigm involving context-dependent stimulus-response mappings. The context is provided by the block $B$, which is essentially a memory that can be sustained over trials. To characterize a specific experiment, we must specify the CPDs in the architecture. These distributions can be entirely determined by the experiment description (embodied in Figure 1B,C). We toss in one twist to the model, which is to incorporate four parameters into the CPDs that permit us to specify aspects of the human cognitive architecture, as follows: , the degree of task knowledge (0: no knowledge; 1: perfect knowledge); , the persistence of the block memory (0: memory decays completely from one trial to the next; 1: memory is perfect); and $_{\text{shape}}$ and $_{\text{color}}$, the rate of transmission of shape and color information between stimulus and category representations. Given these parameters and the experiment description, we can define the CPDs in the model:

$P(B_k = b \mid B_{k-1} = b) = \ _{b\,b} + (1 - )\,N_B$, where is the Kronecker delta and $N_B$ is the number of distinct block (task) identities. This distribution is a mixture of a uniform distribution (no memory of block) and an identity mapping (perfect memory).

$P(C_k^z \mid B_k) = \ P\ (C_k^z \mid B_k) + (1 - )\,N_{C^z}$, where $z \in$ color shape and $N_{C^z}$ is the number of distinct category values along dimension $z$, and $P\ (\ )$ is the probability distribution defined by the experiment and task (see Figure 2B,C). The mixture parameter, , interpolates between a uniform distribution (no knowledge of task) and a distribution that represents complete task knowledge.

$P(R_k \mid B_k\ C_k) = \ P\ (R_k \mid B_k\ C_k) + (1 - )\,N_R$, where $N_R$ is the number of response alternatives (including no-go).

$P(S_k^z = s \mid C_k^z = c\ T = t) \ (1 + \ _z M^z(s\ c))^t$, where $z \in$ color shape and $M^z(s\ c)$ is a membership function that has value 1 if $s$ is an instance of category $c$ along dimension $z$, or 0 otherwise. By this CPD, the normalized probability for stimulus $s$ grows exponentially to

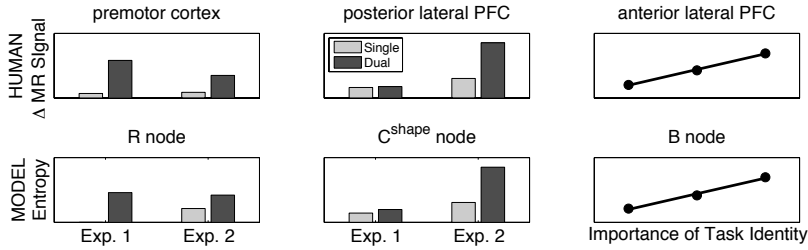

Figure 3: (top row) human neuroimaging data from three brain regions [6], (bottom row) entropy read out from three nodes of the model. Full explanation in the text.

asymptote as a function of time $t$ if $s$ belongs to category $c$, and drops exponentially toward zero if $s$ does not belong to $c$.

This formulation encodes the experiment description—as represented by the $P^*(.)$ probabilities—in the model's CPDs, with smoothing via $\epsilon$ to represent less-than-perfect knowledge of the experiment description.

We would like to read out from the model a response on some trial $k$, given the stimulus on trial $k$, $S_k$, and a history of past stimulus-response pairs, $H_k = \{S_1...S_{k-1}, R_1...R_{k-1}\}$. (In the experiments, subjects are well practiced and make few errors. Therefore, we assume the $R$'s are correct or corrected responses.) The response we wish to read out consists of a choice and the number of time steps required to make the choice. To simulate processing time within a trial, we search over $T$. Larger $T$ correspond to more time for evidence to propagate in the model, which leads to lower entropy distributions over the hidden variables $C_k$ and $R_k$. The model initiates a response when one value of $R_k$ passes a threshold $\theta$, i.e., when $[\max_r P(R_k = r|S_k, T, H_k)] > \theta$. This yields the response time (RT)

$$t^* = \min\left\{t \,\middle|\, \left[\max_r P(R_k = r|S_k, T = t, H_k)\right] > \theta\right\} \tag{1}$$

and the response $r^* = \operatorname{argmax}_r P(R_k = r|S_k, T = t^*, H_k)$.

## 4 Simulation Results

We simulated the model on a trial sequence like that in the human study. We obtained mean RTs and error rates from the model in the four experimental conditions of the two experiments (see the filled cells of Figure 1B,C). The model's five parameters—$\epsilon$, $\lambda$, $\gamma_{\text{shape}}$, $\gamma_{\text{color}}$, and $\theta$—were optimized to obtain the maximum correlation between the mean RTs obtained from the simulation (Equation 1) and the human data (Figure 1D). This optimization resulted in a correlation between human and simulation RTs of 0.99 (compare Figure 1D and E), produced by parameter values $\epsilon = 0.87$, $\lambda = 0.79$, $\gamma_{\text{shape}} = 0.34$, $\gamma_{\text{color}} = 0.88$, and $\theta = 0.63$.

To express simulation time in units of milliseconds— the measure of time collected in the human data—we allowed an affine transform, which includes two free parameters: an *offset* constant indicating the time required for early perceptual and late motor processes, which are not embodied in the model, and a *scale* constant to convert units of simulation time to milliseconds. With these two transformation parameters, the model had a total of seven parameters. The astute reader will note that there are only eight data points to fit, and one should therefore not be impressed by a close match between simulation and data. However, our goal is to constrain model parameters with this fit, and then explore emergent properties of the resulting fully constrained model.

One indication of model robustness is how well the model generalizes to sequences of trials other than the one on which it was optimized. Across 11 additional generalization runs, the correlation between model and empirical data remained high with low variability ($\bar{\rho} = 0.97$, $\sigma_\rho = 0.004$). Another indication of the robustness of the result is to determine how sensitive the model is to the choice of parameters. If randomly selected parameters yield large correlations, then the model architecture itself is responsible for the good fit, not the particular choice of parameters. To perform this test, we excluded parameters ranges in which the model failed to respond reliably (i.e.,

the model never attained the response criterion of Equation 1), or in which the model produced no RT variation across conditions. These requirements led to parameter ranges of: $0.8 \leq \epsilon \leq 0.98$; $0.1 \leq \gamma_{color}, \gamma_{shape} \leq 1.5$; $0.6 \leq \lambda \leq 0.98$; $0.65 \leq \theta \leq 0.85$. All randomly selected combinations of parameters in these ranges led to correlation values greater than 0.9, demonstrating that the qualitative fit between model and behavioral results was insensitive to parameter selection, and that the structure of the model is largely responsible for the fit obtained.

Koechlin, Ody, and Kouneiher [6] collected not only behavioral data, but also neuroimaging data that identified brain regions involved in control, and how these brain regions modulated their activation across experimental manipulations. There were three manipulations in the experiments: (1) the demand on response selection (varied along rows of Figure 1C), (2) the importance of task identity (varied along the three columns of both Figure 1B and 1C), and (3) the demand of stimulus classification and selection of stimulus dimensions (varied along rows of Figure 1B). The top row of Figure 3 shows effects of these experimental manipulations on the fMRI BOLD response of three different brain regions.

The remarkable result obtained in our simulations is that we identified three components of the model that produced signatures analogous to those of the fMRI BOLD response in three cortical areas. We hypothesized that neural (fMRI) activity in the brain might be related to the *entropy* of nodes in the model, on account of the fact that when entropy is high, many possibilities must be simultaneously represented, which may lead to greater BOLD signal. Because fMRI techniques introduce significant blurring in time, any measure in the model corresponding to the fMRI signal would need to be integrated over the time of a trial. We therefore computed the mean entropy of each model node over time $T = 1...t^*$ within a trial. We then averaged the entropy measure across trials within a condition, precisely as we did the RTs. To compare these entropy measures to the imaging data, the value corresponding to the bottom left cell of each experiment array (see Figure 1B and 1C) was subtracted from all of the conditions of that particular experiment. This subtraction was performed because the nature of the MRI signal is relative, and these two cells form the baseline conditions within the empirical observations. After performing this normalization, the values for $R$ and $C^{\mathrm{shape}}$ were then collapsed across the columns in panels B and C of Figure 1, resulting in a bar for each row within each panel. Additionally, the values for $B$ were then collapsed across the rows of each panel, resulting in a value for each column. The model entropy results are shown in the bottom row of Figure 3, and comparison with the top row reveals an exact correspondence. We emphasize that these results are obtained with the model which was fully constrained by fitting the RT data. Thus, these results are emergent properties of the model.

Based on functional neuroanatomy, the correspondence between model components and brain regions is quite natural. Starting with the left column of Figure 3, uncertainty in the model's response corresponds to activity in premotor cortex. This activity is greater when the block calls for two distinct responses than when it calls for one. In the middle column of Figure 3, the uncertainty of shape categorization corresponds to activity in posterior lateral prefrontal cortex. This region is thought to be involved in the selection of task-relevant information, which is consistent with the nature of the current conditions that produce increases. In the right column of Figure 3, the uncertainty of the task identity (block) in the model corresponds to activity in anterior lateral PFC, a brain region near areas known to be involved in WM maintenance. Interestingly, the *lower* the entropy the *higher* the neural activity, in contrast to the other two regions. There is a natural explanation for this inversion, though: entropy is high in the block node when the block representation matters the least, i.e., when the stimulus-response mapping does not depend on knowing the task identity. Thus, higher entropy of the block node actually connotes less information to be maintained due to the functional equivalence among classes.

## 5 Discussion

We proposed a theoretical framework for understanding cognitive control which provides a parsimonious account of behavioral and neuroimaging data from two large experiments. These experiments are sufficiently broad that they subsume several other experimental paradigms (e.g., Stroop, task switching). Koechlin et al. [6] explain their findings in terms of a descriptive model that involves a complex hierarchy of control processes within prefrontal cortex. The explanation for the neuroimaging data that emerges from our model is arguable simpler and more intuitive.

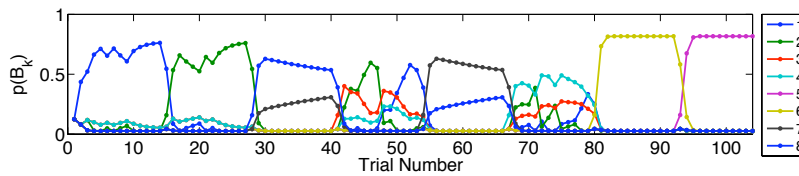

Figure 4: Task (block) representation over a sequence of trials that involves all eight task types.

The key insight that underlies our model is the notion that cortical representations are intrinsically probabilistic. This notion is not too surprising to theorists in computational neuroscience, but it leads to a perspective that is novel within the field of control: that the all-or-none updating of WM can be replaced with a probabilistic notion of updating, and the view that WM holds competing hypotheses in parallel. Framing WM in probabilistic terms also offers a principled explanation for why WM should decay. The parameter $\lambda$ controls a tradeoff between the ability to hold information over time and the ability to update when new relevant information arrives. In contrast, many neural network models have two distinct parameters that control these aspects of memory.

Another novelty of our approach is the notion of that control results from dynamical inference processes, instead of being conceived of as resulting from long-term policy learning. Inference plays a critical role on the WM (task identity) representation: WM is maintained not solely from internal processes (e.g., the recurrent connections in a neural net), but is continually influenced by the ongoing stream of stimuli via inference. The stimulus stream sometimes supports the WM representation and sometimes disrupts it. Figure 4 shows the trial-to-trial dynamics of the WM in our model. Note that depending on the task, the memory looks quite different. When the stimulus-response pairs are ambiguous as to the task, the representation becomes less certain. Fortunately for the model's performance, this is exactly the circumstance in which remembering the task identity is least critical.

Figure 4 also points to a promising future direction for the model. The stream of trials clearly shows strong sequential effects. We are currently pursuing opportunities to examine the model's predictions regarding performance on the first trial in a block versus subsequent trials. The model shows an effect observed in the task switching literature: initial trial performance is poor, but control rapidly tunes to the task and subsequent trials are more efficient and roughly comparable.

Our model seems to have surprisingly strong predictive power. This power comes about from the fact that the model expresses a form of bounded rationality: the model encodes the structure of the task, subject to limitations on memory, learning, and the rate of perceptual processing. Exploiting this bounded rationality leads to strong constraints, few free parameters, and the ability to extend the model to new tasks without introducing additional free parameters.

# References

[1] E. K. Miller and J. D. Cohen. An integrative theory of prefrontal cortex function. *Annual Review of Neuroscience*, 24:167–202, 2001.

[2] A. Newell and H. A. Simon. *Human Problem Solving*. Prentice-Hall, Englewood Cliffs, NJ, 1972.

[3] Robert D. Rogers and Stephen Monsell. Costs of a predictable switch between simple cognitive tasks. *Journal of Experimental Psychology: General*, 124:207–231, 1995.

[4] Nick Yeung and Stephen Monsell. Switching between tasks of unequal familiarity: the role of stimulus-attribute and response-set selection. *J Exp Psychol Hum Percept Perform*, 29(2):455–469, 2003.

[5] C. M. MacLeod. Half a century of research on the Stroop effect: An integrative review. *Psychological Bulletin*, 109:163–203, 1991.

[6] E. Koechlin, C. Ody, and F. Kouneiher. Neuroscience: The architecture of cognitive control in the human prefrontal cortex. *Science*, 424:1181–1184, 2003.

[7] J. D. Cohen, K. Dunbar, and J. L. McClelland. On the control of automatic processes: A parallel distributed processing model of the Stroop effect. *Psychological Review*, 97(3):332–361, 1990.

[8] S. J. Gilbert and T. Shallice. Task switching: A pdp model. *Cognitive Psychology*, 44:297–337, 2002.

[9] N. P. Rougier, D. Noelle, T. S. Braver, J. D. Cohen, and R. C. O'Reilly. Prefrontal cortex and the flexibility of cognitive control: Rules without symbols. *Proceedings of the National Academy of Sciences*, 102(20):7338–7343, 2005.

[10] M. J. Frank and R. C. O'Reilly. A mechanistic account of striatal dopamine function in human cognition: Psychopharmacological studies with cabergoline and haloperidol. *Behavioral Neuroscience*, 120:497–517, 2006.

[11] Stephen Monsell. Control of mental processes. In V. Bruce, editor, *Unsolved mysteries of the mind: Tutorial essays in cognition*, pages 93–148. Psychology press, Hove, UK, 1996.

[12] R S Zemel, P Dayan, and A Pouget. Probabilistic interpretation of population codes. *Neural Comput*, 10(2):403–430, 1998.

